# Pairwise Neural Network Classifiers with Probabilistic Outputs

**David  Price**
A2iA and ESPCI
3 Rue de l'Arrivée, BP 59
75749 Paris Cedex 15, France
a2ia@dialup.francenet.fr

**Stefan  Knerr**
ESPCI and CNRS (UPR A0005)
10, Rue Vauquelin, 75005 Paris, France
knerr@neurones.espci.fr

**Léon  Personnaz,  Gérard  Dreyfus**
ESPCI, Laboratoire d'Electronique
10, Rue Vauquelin, 75005 Paris, France
dreyfus@neurones.espci.fr

## Abstract

Multi-class classification problems can be efficiently solved by partitioning the original problem into sub-problems involving only two classes: for each pair of classes, a (potentially small) neural network is trained using only the data of these two classes. We show how to combine the outputs of the two-class neural networks in order to obtain posterior probabilities for the class decisions. The resulting probabilistic pairwise classifier is part of a handwriting recognition system which is currently applied to check reading. We present results on real world data bases and show that, from a practical point of view, these results compare favorably to other neural network approaches.

## 1   Introduction

Generally, a pattern classifier consists of two main parts: a feature extractor and a classification algorithm. Both parts have the same ultimate goal, namely to transform a given input pattern into a representation that is easily interpretable as a class decision. In the case of feedforward neural networks, the interpretation is particularly easy if each class is represented by one output unit. For many pattern recognition problems, it suffices that the classifier compute the class of the input pattern, in which case it is common practice to associate the pattern to the class corresponding to the maximum output of the classifier. Other problems require graded (soft) decisions, such as probabilities, at the output of the

classifier for further use in higher context levels: in speech or character recognition for instance, the probabilistic outputs of the phoneme (character) recognizer are often used by a Hidden-Markov-Model algorithm or by some other dynamic programming algorithm to compute the most probable word hypothesis.

In the context of classification, it has been shown that the minimization of the Mean Square Error (MSE) yields estimates of a posteriori class probabilities [Bourlard & Wellekens, 1990; Duda & Hart, 1973]. The minimization can be performed by a feedforward multilayer perceptrons (MLP's) using the backpropagation algorithm, which is one of the reasons why MLP's are widely used for pattern recognition tasks. However, MLPs have well-known limitations when coping with real-world problems, namely long training times and unknown architecture.

In the present paper, we show that the estimation of posterior probabilities for a K-class problem can be performed efficiently using estimates of posterior probabilities for K(K-1)/2 two-class sub-problems. Since the number of sub-problems increases as $K^2$, this procedure was originally intended for applications involving a relatively small number of classes, such as the 10 classes for the recognition of handwritten digits [Knerr et al., 1992]. In this paper we show that this approach is also viable for applications with $K \gg 10$.

The probabilistic pairwise classifier presented in this paper is part of a handwriting recognition system, discussed elsewhere [Simon, 1992], which is currently applied to check reading. The purpose of our character recognizer is to classify pre-segmented characters from cursive handwriting. The probabilistic outputs of the recognizer are used to estimate word probabilities. We present results on real world data involving 27 classes, compare these results to other neural network approaches, and show that our probabilistic pairwise classifier is a powerful tool for computing posterior class probabilities in pattern recognition problems.

## 2    Probabilistic Outputs from Two-class Classifiers

Multi-class classification problems can be efficiently solved by "divide and conquer" strategies which partition the original problem into a set of K(K-1)/2 two-class problems. For each pair of classes $\omega_i$ and $\omega_j$, a (potentially small) neural network with a single output unit is trained on the data of the two classes [Knerr et al., 1990, and references therein]. In this section, we show how to obtain probabilistic outputs from each of the two-class classifiers in the pairwise neural network classifier (Figure 1).

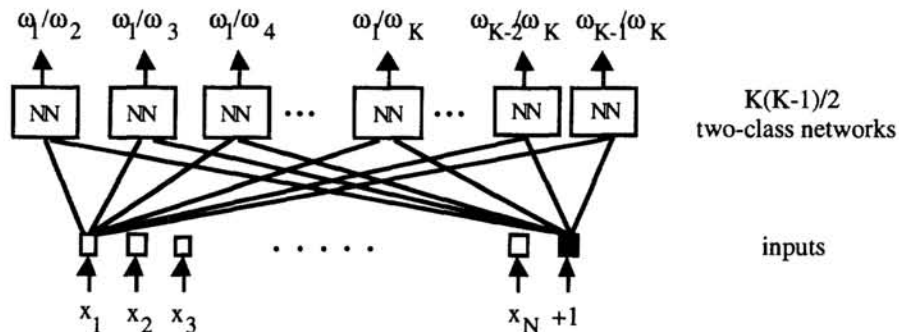

Figure 1: Pairwise neural network classifier.

It has been shown that the minimization of the MSE cost function (or likewise a cost function based on an entropy measure, [Bridle, 1990]) leads to estimates of posterior probabilities. Of course, the quality of the estimates depends on the number and distribution of examples in the training set and on the minimization method used.

In the theoretical case of two classes $\omega_1$ and $\omega_2$, each Gaussian distributed, with means $\mathbf{m}_1$ and $\mathbf{m}_2$, a priori probabilities $Pr_1$ and $Pr_2$, and equal covariance matrices $\Sigma$, the posterior probability of class $\omega_1$ given the pattern $\mathbf{x}$ is:

$$Pr(class=\omega_1 \mid X=\mathbf{x}) = \frac{1}{1 + \frac{Pr_2}{Pr_1} \exp\left(-\frac{1}{2}\left(2\mathbf{x}^T\Sigma^{-1}(\mathbf{m}_1-\mathbf{m}_2) + \mathbf{m}_2^T\Sigma^{-1}\mathbf{m}_2 - \mathbf{m}_1^T\Sigma^{-1}\mathbf{m}_1\right)\right)} \quad (1)$$

Thus a single neuron with a sigmoidal transfer function can compute the posterior probabilities for the two classes.

However, in the case of real world data bases, classes are not necessarily Gaussian distributed, and therefore the transformation of the $K(K-1)/2$ outputs of our pairwise neural network classifier to posterior probabilities proceeds in two steps.

In the first step, a class-conditional probability density estimation is performed on the linear output of each two-class neural network: for both classes $\omega_i$ and $\omega_j$ of a given two-class neural network, we fit the probability density over $v_{ij}$ (the weighted sum of the inputs of the output neuron) to a function. We denote by $\omega_{ij}$ the union of classes $\omega_i$ and $\omega_j$. The resulting class-conditional densities $p(v_{ij} \mid \omega_i)$ and $p(v_{ij} \mid \omega_j)$ can be transformed to probabilities $Pr(\omega_i \mid \omega_{ij} \wedge (V_{ij}=v_{ij}))$ and $Pr(\omega_j \mid \omega_{ij} \wedge (V_{ij}=v_{ij}))$ via the Bayes rule (note that $Pr(\omega_{ij} \wedge (V_{ij}=v_{ij}) \mid \omega_i) = Pr((V_{ij}=v_{ij}) \mid \omega_i))$:

$$Pr(\omega_i \mid \omega_{ij}\wedge(V_{ij}=v_{ij})) = \frac{p(v_{ij} \mid \omega_i)\, Pr(\omega_i)}{\sum_{k\in\{i,j\}} p(v_{ij} \mid \omega_k)\, Pr(\omega_k)} \quad (2)$$

It is a central assumption of our approach that the linear classifier output $v_{ij}$ is as informative as the input vector $\mathbf{x}$. Hence, we approximate $Pr_{ij} = Pr(\omega_i \mid \omega_{ij} \wedge (X=\mathbf{x}))$ by $Pr(\omega_i \mid \omega_{ij} \wedge (V=v_{ij}))$. Note that $P_{ji} = 1-P_{ij}$.

In the second step, the probabilities $Pr_{ij}$ are combined to obtain posterior probabilities $Pr(\omega_i \mid (X=\mathbf{x}))$ for all classes $\omega_i$ given a pattern $\mathbf{x}$. Thus, the network can be considered as generating an intermediate data representation in the recognition chain, subject to further processing [Denker & LeCun, 1991]. In other words, the neural network becomes part of the preprocessing and contributes to dimensionality reduction.

## 3   Combining the Probabilities $Pr_{ij}$ of the Two-class Classifiers to a posteriori Probabilities

The set of two-class neural network classifiers discussed in the previous section results in probabilities $Pr_{ij}$ for all pairs $(i, j)$ with $i \neq j$. Here, the task is to express the posterior probabilities $Pr(\omega_i \mid (X=\mathbf{x}))$ as functions of the $Pr_{ij}$.

We assume that each pattern belongs to only one class:

$$\Pr\left(\bigcup_{j=1}^{K} \omega_j \mid (X=\mathbf{x})\right) = 1 \tag{3}$$

From the definition of $\omega_{ij}$, it follows for any given i:

$$\Pr\left(\bigcup_{j=1}^{K} \omega_j \mid (X=\mathbf{x})\right) = \Pr\left(\bigcup_{j=1,\ j\neq i}^{K} \omega_{ij} \mid (X=\mathbf{x})\right) = 1 \tag{4}$$

Using the closed form expression for the probability of the union of N events $E_i$:

$$\Pr\left(\bigcup_{i=1}^{N} E_i\right) = \sum_{i=1}^{N} \Pr(E_i) + ... + (-1)^{k-1} \sum_{i_1<...<i_k}^{N} \Pr(E_{i_1}\wedge...\wedge E_{i_k}) + ... + (-1)^{N-1} \Pr(E_1\wedge...\wedge E_N)$$

it follows from (4):

$$\sum_{j=1,j\neq i}^{K} \Pr(\omega_{ij} \mid (X=\mathbf{x})) - (K-2)\, \Pr(\omega_i \mid (X=\mathbf{x})) = 1 \tag{5}$$

With

$$\Pr_{ij} = \Pr(\omega_i \mid \omega_{ij}\wedge(X=\mathbf{x})) = \frac{\Pr(\omega_i\wedge\omega_{ij}\wedge(X=\mathbf{x}))}{\Pr(\omega_{ij}\wedge(X=\mathbf{x}))} = \frac{\Pr(\omega_i \mid (X=\mathbf{x}))}{\Pr(\omega_{ij} \mid (X=\mathbf{x}))} \tag{6}$$

one obtains the final expression for the K posterior probabilities given the K(K-1)/2 two-class probabilities $\Pr_{ji}$ :

$$\Pr(\omega_i \mid (X=\mathbf{x})) = \frac{1}{\displaystyle\sum_{j=1,j\neq i}^{K} \frac{1}{\Pr_{ij}} - (K-2)} \tag{7}$$

In [Refregier et al., 1991], a method was derived which allows to compute the K posterior probabilities from only (K-1) two-class probabilities using the following relation between posterior probabilities and two-class probabilities:

$$\frac{\Pr_{ij}}{\Pr_{ji}} = \frac{\Pr(\omega_i \mid (X=\mathbf{x}))}{\Pr(\omega_j \mid (X=\mathbf{x}))} \tag{8}$$

However, this approach has several practical drawbacks. For instance, in practice, the quality of the estimation of the posterior probabilities depends critically on the choice of the set of (K-1) two-class probabilities, and finding the optimal subset of (K-1) $\Pr_{ij}$ is costly, since it has to be performed for each pattern at recognition time.

## 4   Application to Cursive Handwriting Recognition

We applied the concepts described in the previous sections to the classification of pre-segmented characters from cursive words originating from real-world French postal checks. For cursive word recognition it is important to obtain probabilities at the output of the character classifier since it is necessary to establish an ordered list of hypotheses along with a confidence value for further processing at the word recognition level: the probabilities can be passed to an Edit Distance algorithm [Wagner et al., 1974] or to a Hidden-Markov-Model algorithm [Kundu et al., 1989] in order to compute recognition scores for words. For the recognition of the amounts on French postal checks we used an Edit Distance algorithm and made extensive use of the fact that we are dealing with a limited vocabulary (28 words).
The 27 character classes are particularly chosen for this task and include pairs of letters such as "fr", "gt", and "tr" because these combinations of letters are often difficult to pre-segment. Other characters, such as "k" and "y" are not included because they do not appear in the given 28 word vocabulary.

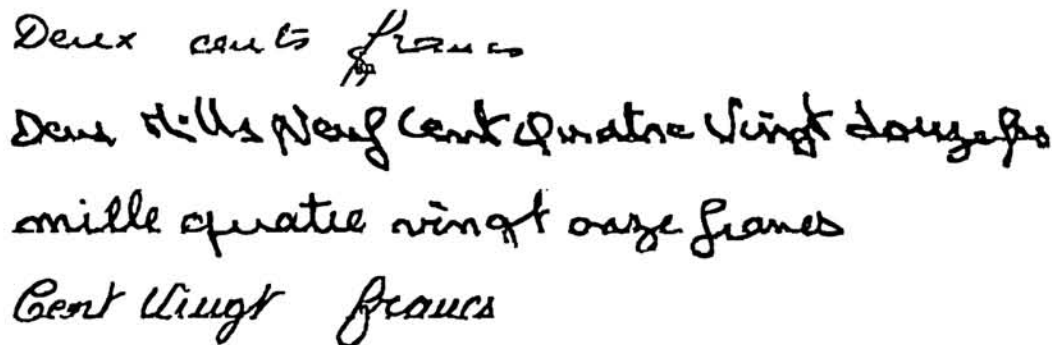

Figure 2: Some examples of literal amounts from live French postal checks.

A data base of about 3,300 literal amounts from postal checks (approximately 16,000 words) was annotated and, based on this annotation, segmented into words and letters using heuristic methods [Simon et al., 1994]. Figure 2 shows some examples of literal amounts. The writing styles vary strongly throughout the data base and many checks are difficult to read even for humans. Note that the images of the pre-segmented letters may still contain some of the ligatures or other extraneous parts and do not in general resemble hand-printed letters. The total of about 55,000 characters was divided into three sets: training set (20,000), validation set (20,000), and test set (15,000). All three sets were used without any further data base cleaning. Therefore, many letters are not only of very bad quality, but they are truly ambiguous: it is not possible to recognize them uniquely without word context.

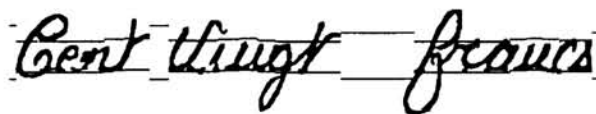

Figure 3: Reference lines indicating upper and lower limit of lower case letters.

Before segmentation, two reference lines were detected for each check (Figure 3). They indicate an estimated upper and lower limit of the lower case letters and are used for

normalization of the pre-segmented characters (Figure 4) to 10 by 24 pixel matrices with 16 gray values (Figure 5). This is the representation used as input to the classifiers.

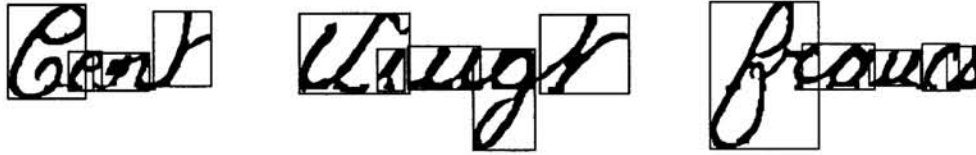

Figure 4: Segmentation of words into isolated letters (ligatures are removed later).

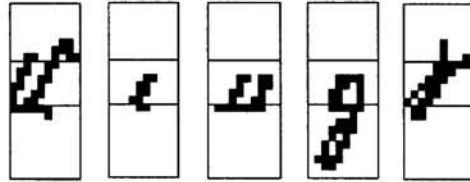

Figure 5: Size normalized letters: 10 by 24 pixel matrices with 16 gray values.

The simplest two-class classifier is a single neuron; thus, 351 neurons of the resulting pairwise classifier were trained on the training data using the generalized delta rule (sigmoidal transfer function). In order to avoid overfitting, training was stopped at the minimum of MSE on the validation set. The probability densities $p(v_{ij} \mid \omega_i)$ were estimated on the validation set: for both classes $\omega_i$ and $\omega_j$ of a given neuron, we fitted the probability densities over the linear output $v_{ij}$ to a Gaussian. The two-class probabilities $Pr_{ij}$ and $Pr_{ji}$ were then obtained via Bayes rule. The 351 probabilities $Pr_{ij}$ were combined using equation (7) in order to obtain a posteriori probabilities $Pr(\omega_i \mid (X=x))$, $i \in \{1,..,27\}$. However, the a priori probabilities for letters as given by the training set are different from the prior probabilities in a given word context [Bourlard & Morgan, 1994]. Therefore, we computed the posterior probabilities either by using, in Bayes rule, the prior probabilities of the letters in the training set, or by assuming that the prior probabilities are equal. In the first case, many informative letters, for instance those having ascenders or descenders, have little chance to be recognized at all due to small a priori probabilities.

Table 1 gives the recognition performances on the test set for classes assumed to have equal a priori probabilities as well as for the true a priori probabilities of the test set. For each pattern, an ordered list (in descending order) of posterior class probabilities was generated; the recognition performance is given (i) in terms of percentage of true classes found in first position, and (ii) in terms of average position of the true class in the ordered list. As mentioned above, the results of the first column are the most relevant ones, since the classifier outputs are subsequently used for word recognition. Note that the recognition rate (first position) of isolated letters without context for a human reader can be estimated to be around 70% to 80%.

We compared the results of the pairwise classifier to a number of other neural network classification algorithms. First, we trained MLPs with one and two hidden layers and various numbers of hidden units using stochastic backpropagation. Here again, training was stopped based on the minimum MSE on the validation set. Second, we trained MLPs with a single hidden layer using the Softmax training algorithm [Bridle, 1990]. As a third approach, we trained 27 MLPs with 10 hidden units each, each MLP separating one class from all others. Table 1 gives the recognition performances on the test set. The Softmax

training algorithm clearly gives the best results in terms of recognition performance. However, the pairwise classifier has three very attractive features for classifier design:
(i) training is faster than for MLP's by more than one order of magnitude; therefore, many different designs (changing pattern representations for instance) can be tested at a small computational cost;
(ii) in the same spirit, adding a new class or modifying the training set of an existing one can be done without retraining all two-class classifiers;
(iii) at least as importantly, the procedure gives more insight into the classification problem than MLP's do.

| Classifier | AveragePosition equal prior probs | First Position equal prior probs | AveragePosition true prior probs | First Position true prior probs |
|---|---|---|---|---|
| Pairwise Classifier | 2.9 | 48.9 % | 2.6 | 52.2 % |
| MLP (100 hid. units) | 3.6 | 48.9 % | 2.7 | 60.0 % |
| Softmax (100 hid. units) | 2.6 | 54.9 % | 2.2 | 61.9 % |
| 27 MLPs | 3.2 | 41.6 % | 2.4 | 55.8 % |

Table 1: Recognition performances on the test set in terms of average position and recognition rate (first position) for the various neural networks used.

Our pairwise classifier is part of a handwriting recognition system which is currently applied to check reading. The complete system also incorporates other character recognition algorithms as well as a word recognizer which operates without pre-segmentation. The result of the complete check recognition chain on a set of test checks is the following: (i) at the word level, 83.3% of true words are found in first position; (ii) 64.1% of well recognized literal amounts are found in first position [Simon et al., 1994]. Recognizing also the numeral amount, we obtained 80% well recognized checks for 1% error.

## 5 Conclusion

We have shown how to obtain posterior class probabilities from a set of pairwise classifiers by (i) performing class density estimations on the network outputs and using Bayes rule, and (ii) combining the resulting two-class probabilities. The application of our pairwise classifier to the recognition of real world French postal checks shows that the procedure is a valuable tool for designing a recognizer, experimenting with various data representations at a small computational cost and, generally, getting insight into the classification problem.

### Acknowledgments

The authors wish to thank J.C. Simon, N. Gorsky, O. Baret, and J.C. Deledicq for many informative and stimulating discussions.

## References

H.A. Bourlard, N. Morgan (1994). *Connectionist Speech Recognition.* Kluwer Academic Publishers.

H.A. Bourlard, C. Wellekens (1990). Links between Markov Models and Multilayer Perceptrons. IEEE Transactions on Pattern Analysis and Machine Intelligence, Vol. 12, No. 12, 1167-1178.

J.S. Bridle (1990). Probabilistic Interpretation of Feedforward Classification Network Outputs, with Relationships to Statistical Pattern Recognition. In *Neurocomputing: Algorithms, Architectures and Applications*, Fogelman-Soulie, and Herault (eds.). NATO ASI Series, Springer.

J.S. Denker, Y.LeCun (1991). Transforming Neural-Net Output Levels to Probability Distributions. In *Advances in Neural Information Processing Systems 3*, Lippmann, Moody, Touretzky (eds.). Morgan Kaufman.

R.O. Duda, P.E. Hart (1973). *Pattern Classification and Scene Analysis.* Wiley.

S. Knerr, L. Personnaz, G. Dreyfus (1990). Single-Layer Learning Revisited: A Stepwise Procedure for Building and Training a Neural Network. In *Neurocomputing: Algorithms, Architectures and Applications*, Fogelman-Soulie and Herault (eds.). NATO ASI Series, Springer.

S. Knerr, L. Personnaz, G. Dreyfus (1992). Handwritten Digit Recognition by Neural Networks with Single-Layer Training. *IEEE Transactions on Neural Networks*, Vol. 3, No. 6, 962-968.

A. Kundu, Y. He, P. Bahl (1989). Recognition of Handwritten Words: First and Second Order Hidden Markov Model Based Approach. *Pattern Recognition*, Vol. 22, No.3.

J.C. Simon (1992). Off-Line Cursive Word Recognition. *Proceedings of the IEEE*, Vol. 80, No. 7, 1150-1161.

Ph. Refregier, F. Vallet (1991). Probabilistic Approach for Multiclass Classification with Neural Networks. *Int. Conference on Artificial Networks*, Vol. 2, 1003-1007.

J.C. Simon, O. Baret, N. Gorski (1994). Reconnaisance d'écriture manuscrite. *Compte Rendu Academie des Sciences*, Paris, t. 318, Serie II, 745-752.

R.A. Wagner, M.J. Fisher (1974). The String to String Correction Problem. J.A.C.M. Vol. 21, No. 5, 168-173.
